# The Kernel Trick for Distances

**Bernhard Schölkopf**
Microsoft Research
1 Guildhall Street
Cambridge, UK
*bs@kyb.tuebingen.mpg.de*

## Abstract

A method is described which, like the kernel trick in support vector machines (SVMs), lets us generalize distance-based algorithms to operate in feature spaces, usually nonlinearly related to the input space. This is done by identifying a class of kernels which can be represented as norm-based distances in Hilbert spaces. It turns out that common kernel algorithms, such as SVMs and kernel PCA, are actually really distance based algorithms and can be run with that class of kernels, too.

As well as providing a useful new insight into how these algorithms work, the present work can form the basis for conceiving new algorithms.

## 1 Introduction

One of the crucial ingredients of SVMs is the so-called kernel trick for the computation of dot products in high-dimensional feature spaces using simple functions defined on pairs of input patterns. This trick allows the formulation of nonlinear variants of any algorithm that can be cast in terms of dot products, SVMs being but the most prominent example [13, 8]. Although the mathematical result underlying the kernel trick is almost a century old [6], it was only much later [1, 3, 13] that it was made fruitful for the machine learning community. Kernel methods have since led to interesting generalizations of learning algorithms and to successful real-world applications. The present paper attempts to extend the utility of the kernel trick by looking at the problem of which kernels can be used to compute *distances* in feature spaces. Again, the underlying mathematical results, mainly due to Schoenberg, have been known for a while [7]; some of them have already attracted interest in the kernel methods community in various contexts [11, 5, 15].

Let us consider training data $(x_1, y_1), \ldots, (x_m, y_m) \in \mathcal{X} \times \mathcal{Y}$. Here, $\mathcal{Y}$ is the set of possible outputs (e.g., in pattern recognition, $\{\pm 1\}$), and $\mathcal{X}$ is some nonempty set (the domain) that the patterns are taken from. We are interested in predicting the outputs $y$ for previously unseen patterns $x$. This is only possible if we have some measure that tells us how $(x, y)$ is related to the training examples. For many problems, the following approach works: informally, we want similar inputs to lead to similar outputs. To formalize this, we have to state what we mean by *similar*. On the outputs, similarity is usually measured in terms of a *loss function*. For instance, in the case of pattern recognition, the situation is simple: two outputs can either be identical or different. On the inputs, the notion of similarity is more complex. It hinges on a representation of the patterns and a suitable similarity measure operating on that representation.

One particularly simple yet surprisingly useful notion of (dis)similarity — the one we will use in this paper — derives from embedding the data into a Euclidean space and utilizing geometrical concepts. For instance, in SVMs, similarity is measured by dot products (i.e. angles and lengths) in some high-dimensional feature space $F$. Formally, the patterns are first mapped into $F$ using $\phi : \mathcal{X} \to F$, $x \mapsto \phi(x)$, and then compared using a dot product $\langle \phi(x), \phi(x') \rangle$. To avoid working in the potentially high-dimensional space $F$, one tries to pick a feature space in which the dot product can be evaluated directly using a nonlinear function in input space, i.e. by means of the *kernel trick*

$$k(x, x') = \langle \phi(x), \phi(x') \rangle. \tag{1}$$

Often, one simply chooses a kernel $k$ with the property that there exists some $\phi$ such that the above holds true, without necessarily worrying about the actual form of $\phi$ — already the *existence* of the linear space $F$ facilitates a number of algorithmic and theoretical issues. It is well established that (1) works out for *Mercer* kernels [3, 13], or, equivalently, positive definite kernels [2, 14]. Here and below, indices $i$ and $j$ by default run over $1, \ldots, m$.

**Definition 1 (Positive definite kernel)** *A symmetric function* $k : \mathcal{X} \times \mathcal{X} \to \mathbb{R}$ *which for all* $m \in \mathbb{N}, x_i \in \mathcal{X}$ *gives rise to a positive definite Gram matrix, i.e. for which for all* $c_i \in \mathbb{R}$ *we have*

$$\sum_{i,j=1}^{m} c_i c_j K_{ij} \geq 0, \quad where\ K_{ij} := k(x_i, x_j), \tag{2}$$

*is called a* positive definite (pd) kernel.

One particularly intuitive way to construct a feature map satisfying (1) for such a kernel $k$ proceeds, in a nutshell, as follows (for details, see [2]):

*1. Define a feature map*

$$\phi : \mathcal{X} \to \mathbb{R}^{\mathcal{X}}, \quad x \mapsto k(., x). \tag{3}$$

Here, $\mathbb{R}^{\mathcal{X}}$ denotes the space of functions mapping $\mathcal{X}$ into $\mathbb{R}$.

*2. Turn it into a linear space* by forming linear combinations

$$f(.) = \sum_{i=1}^{m} \alpha_i k(., x_i), \quad g(.) = \sum_{j=1}^{m'} \beta_j k(., x_j'), \quad (m, m' \in \mathbb{N}, \alpha_i, \beta_j \in \mathbb{R}, x_i, x_j' \in \mathcal{X}). \tag{4}$$

*3. Endow it with a dot product* $\langle f, g \rangle := \sum_{i=1}^{m} \sum_{j=1}^{m'} \alpha_i \beta_j k(x_i, x_j')$, and turn it into a Hilbert space $H_k$ by completing it in the corresponding norm.

Note that in particular, by definition of the dot product, $\langle k(., x), k(., x') \rangle = k(x, x')$, hence, in view of (3), we have $k(x, x') = \langle \phi(x), \phi(x') \rangle$, the kernel trick. This shows that pd kernels can be thought of as (nonlinear) generalizations of one of the simplest *similarity measures*, the canonical dot product $\langle x, x' \rangle$, $x, x' \in \mathbb{R}^N$. The question arises as to whether there also exist generalizations of the simplest *dissimilarity* measure, the distance $\|x - x'\|^2$.

Clearly, the distance $\|\phi(x) - \phi(x')\|^2$ in the feature space associated with a pd kernel $k$ can be computed using the kernel trick (1) as $k(x, x) + k(x', x') - 2k(x, x')$. Positive definite kernels are, however, not the full story: there exists a *larger* class of kernels that can be used as generalized distances, and the following section will describe why.

## 2   Kernels as Generalized Distance Measures

Let us start by considering how a dot product and the corresponding distance measure are affected by a translation of the data, $x \mapsto x - x_0$. Clearly, $\|x - x'\|^2$ is translation invariant

while $\langle x, x' \rangle$ is not. A short calculation shows that the effect of the translation can be expressed in terms of $\|. - .\|^2$ as

$$\langle (x - x_0), (x' - x_0) \rangle = \frac{1}{2} \left( -\|x - x'\|^2 + \|x - x_0\|^2 + \|x_0 - x'\|^2 \right). \tag{5}$$

Note that this is, just like $\langle x, x' \rangle$, still a pd kernel: $\sum_{i,j} c_i c_j \langle (x_i - x_0), (x_j - x_0) \rangle = \|\sum_i c_i (x_i - x_0)\|^2 \geq 0$. For any choice of $x_0 \in \mathcal{X}$, we thus get a similarity measure (5) associated with the dissimilarity measure $\|x - x'\|$.

This naturally leads to the question whether (5) might suggest a connection that holds true also in more general cases: what kind of nonlinear dissimilarity measure do we have to substitute instead of $\|. - .\|^2$ on the right hand side of (5) to ensure that the left hand side becomes positive definite? The answer is given by a known result. To state it, we first need to define the appropriate class of kernels.

**Definition 2 (Conditionally positive definite kernel)** *A symmetric function* $k : \mathcal{X} \times \mathcal{X} \rightarrow \mathbb{R}$ *which satisfies (2) for all* $m \in \mathbb{N}$, $x_i \in \mathcal{X}$ *and for all* $c_i \in \mathbb{R}$ *with*

$$\sum_{i=1}^m c_i = 0, \tag{6}$$

*is called a* conditionally positive definite (cpd) kernel.

**Proposition 3 (Connection pd — cpd [2])** *Let* $x_0 \in \mathcal{X}$, *and let* $k$ *be a symmetric kernel on* $\mathcal{X} \times \mathcal{X}$. *Then*

$$\tilde{k}(x, x') := \frac{1}{2}(k(x, x') - k(x, x_0) - k(x_0, x') + k(x_0, x_0)) \tag{7}$$

*is positive definite if and only if* $k$ *is conditionally positive definite.*

The proof follows directly from the definitions and can be found in [2].

This result does generalize (5): the negative squared distance kernel is indeed cpd, for $\sum_i c_i = 0$ implies $-\sum_{i,j} c_i c_j \|x_i - x_j\|^2 = -\sum_i c_i \sum_j c_j \|x_j\|^2 - \sum_j c_j \sum_i c_i \|x_i\|^2 + 2 \sum_{i,j} c_i c_j \langle x_i, x_j \rangle = 2 \sum_{i,j} c_i c_j \langle x_i, x_j \rangle = 2\|\sum_i c_i x_i\|^2 \geq 0$. In fact, this implies that all kernels of the form

$$k(x, x') = -\|x - x'\|^\beta, 0 < \beta \leq 2 \tag{8}$$

are cpd (they are not pd), by application of the following result:

**Proposition 4 ([2])** *If* $k : \mathcal{X} \times \mathcal{X} \rightarrow ]-\infty, 0]$ *is cpd, then so are* $-(-k)^\alpha$ $(0 < \alpha < 1)$ *and* $-\log(1 - k)$.

To state another class of cpd kernels that are not pd, note first that as trivial consequences of Definition 2, we know that (i) sums of cpd kernels are cpd, and (ii) any constant $b \in \mathbb{R}$ is cpd. Therefore, any kernel of the form $k + b$, where $k$ is cpd and $b \in \mathbb{R}$, is also cpd. In particular, since pd kernels are cpd, we can take any pd kernel and offset it by $b$ and it will still be at least cpd. For further examples of cpd kernels, cf. [2, 14, 4, 11].

We now return to the main flow of the argument. Proposition 3 allows us to construct the feature map for $k$ from that of the pd kernel $\tilde{k}$. To this end, fix $x_0 \in \mathcal{X}$ and define $\tilde{k}$ according to (7). Due to Proposition 3, $\tilde{k}$ is positive definite. Therefore, we may employ the Hilbert space representation $\phi : \mathcal{X} \rightarrow H$ of $\tilde{k}$ (cf. (1)), satisfying $\langle \phi(x), \phi(x') \rangle = \tilde{k}(x, x')$, hence

$$\|\phi(x) - \phi(x')\|^2 = \langle \phi(x) - \phi(x'), \phi(x) - \phi(x') \rangle = \tilde{k}(x, x) + \tilde{k}(x', x') - 2\tilde{k}(x, x'). \tag{9}$$

Substituting (7) yields

$$\|\phi(x) - \phi(x')\|^2 = -k(x, x') + \frac{1}{2}\left(k(x, x) + k(x', x')\right). \tag{10}$$

We thus have proven the following result.

**Proposition 5 (Hilbert space representation of cpd kernels [7, 2])** *Let $k$ be a real-valued conditionally positive definite kernel on $\mathcal{X}$, satisfying $k(x, x) = 0$ for all $x \in \mathcal{X}$. Then there exists a Hilbert space $H$ of real-valued functions on $\mathcal{X}$, and a mapping $\phi : \mathcal{X} \to H$, such that*

$$\|\phi(x) - \phi(x')\|^2 = -k(x, x'). \tag{11}$$

*If we drop the assumption $k(x, x) = 0$, the Hilbert space representation reads*

$$\|\phi(x) - \phi(x')\|^2 = -k(x, x') + \frac{1}{2}\left(k(x, x) + k(x', x')\right). \tag{12}$$

It can be shown that if $k(x, x) = 0$ for all $x \in \mathcal{X}$, then $d(x, x') := \sqrt{-k(x, x')} = \|\phi(x) - \phi(x')\|$ is a semi-metric; it is a metric if $k(x, x') \neq 0$ for $x \neq x'$ [2].

We next show how to represent *general* symmetric kernels (thus in particular cpd kernels) as symmetric bilinear forms $Q$ in feature spaces. This generalization of the previously known feature space representation for pd kernels comes at a cost: $Q$ will no longer be a dot product. For our purposes, we can get away with this. The result will give us an intuitive understanding of Proposition 3: we can then write $\tilde{k}$ as $\tilde{k}(x, x') := Q(\phi(x) - \phi(x_0), \phi(x') - \phi(x_0))$. Proposition 3 thus essentially adds an origin in feature space which corresponds to the image $\phi(x_0)$ of one point $x_0$ under the feature map. For translation invariant algorithms, we are always allowed to do this, and thus turn a cpd kernel into a pd one — in this sense, cpd kernels are "as good as" pd kernels.

**Proposition 6 (Vector space representation of symmetric kernels)** *Let $k$ be a real-valued symmetric kernel on $\mathcal{X}$. Then there exists a linear space $H$ of real-valued functions on $\mathcal{X}$, endowed with a symmetric bilinear form $Q(.,.)$, and a mapping $\phi : \mathcal{X} \to H$, such that*

$$k(x, x') = Q(\phi(x), \phi(x')). \tag{13}$$

**Proof** The proof is a direct modification of the pd case. We use the map (3) and linearly complete the image as in (4). Define $Q(f, g) := \sum_{i=1}^{m} \sum_{j=1}^{m'} \alpha_i \beta_j k(x_i, x_j')$. To see that it is well-defined, although it explicitly contains the expansion coefficients (which need not be unique), note that $Q(f, g) = \sum_{j=1}^{m'} \beta_j f(x_j')$, independent of the $\alpha_i$. Similarly, for $g$, note that $Q(f, g) = \sum_i \alpha_i g(x_i)$, hence it is independent of $\beta_j$. The last two equations also show that $Q$ is bilinear; clearly, it is symmetric. ∎

Note, moreover, that by definition of $Q$, $k$ is a reproducing kernel for the feature space (which is not a Hilbert space): for all functions $f$ (4), we have $Q(k(., x), f) = f(x)$; in particular, $Q(k(., x), k(., x')) = k(x, x')$.

Rewriting $\tilde{k}$ as $\tilde{k}(x, x') := Q(\phi(x) - \phi(x_0), \phi(x') - \phi(x_0))$ suggests an immediate generalization of Proposition 3: in practice, we might want to choose other points as origins in feature space — points that do not have a preimage $x_0$ in input space, such as (usually) the mean of a set of points (cf. [12]). This will be useful when considering kernel PCA. Crucial is only that our reference point's behaviour under translations is identical to that of individual points. This is taken care of by the constraint on the sum of the $c_i$ in the following proposition. The asterisk denotes the complex conjugated transpose.

**Proposition 7 (Exercise 2.23, [2])** *Let $K$ be a symmetric matrix, $\mathbf{e} \in \mathbb{R}^m$ be the vector of all ones, $I$ the $m \times m$ identity matrix, and let $\mathbf{c} \in \mathbb{C}^m$ satisfy $\mathbf{e}^*\mathbf{c} = 1$. Then*

$$\tilde{K} := (I - \mathbf{e}\mathbf{c}^*)K(I - \mathbf{c}\mathbf{e}^*) \tag{14}$$

*is positive definite if and only if $K$ is conditionally positive definite.*

**Proof**

"$\Longrightarrow$": suppose $\tilde{K}$ is positive definite, i.e. for any $\mathbf{a} \in \mathbb{C}^m$, we have

$$0 \leq \mathbf{a}^*\tilde{K}\mathbf{a} = \mathbf{a}^*K\mathbf{a} + \mathbf{a}^*\mathbf{e}\mathbf{c}^*K\mathbf{c}\mathbf{e}^*\mathbf{a} - \mathbf{a}^*K\mathbf{c}\mathbf{e}^*\mathbf{a} - \mathbf{a}^*\mathbf{e}\mathbf{c}^*K\mathbf{a}. \tag{15}$$

In the case $\mathbf{a}^*\mathbf{e} = \mathbf{e}^*\mathbf{a} = 0$ (cf. (6)), the three last terms vanish, i.e. $0 \leq \mathbf{a}^*K\mathbf{a}$, proving that $K$ is conditionally positive definite.

"$\Longleftarrow$": suppose $K$ is conditionally positive definite. The map $(I - \mathbf{c}\mathbf{e}^*)$ has its range in the orthogonal complement of $\mathbf{e}$, which can be seen by computing, for any $\mathbf{a} \in \mathbb{C}^m$,

$$\mathbf{e}^*(I - \mathbf{c}\mathbf{e}^*)\mathbf{a} = \mathbf{e}^*\mathbf{a} - \mathbf{e}^*\mathbf{c}\mathbf{e}^*\mathbf{a} = 0. \tag{16}$$

Moreover, being symmetric and satisfying $(I - \mathbf{c}\mathbf{e}^*)^2 = (I - \mathbf{c}\mathbf{e}^*)$, the map $(I - \mathbf{c}\mathbf{e}^*)$ is a projection. Thus $\tilde{K}$ is the restriction of $K$ to the orthogonal complement of $\mathbf{e}$, and by definition of conditional positive definiteness, that is precisely the space where $K$ is positive definite.

■

This result directly implies a corresponding generalization of Proposition 3:

**Proposition 8 (Adding a general origin)** *Let $k$ be a symmetric kernel, $x_1, \ldots, x_m \in \mathcal{X}$, and let $c_i \in \mathbb{C}$ satisfy $\sum_{i=1}^m c_i = 1$. Then*

$$\tilde{k}(x, x') := \frac{1}{2}\left( k(x, x') - \sum_{i=1}^m c_i k(x, x_i) - \sum_{i=1}^m c_i k(x_i, x') + \sum_{i,j=1}^m c_i c_j k(x_i, x_j) \right) \tag{17}$$

*is positive definite if and only if $k$ is conditionally positive definite.*

**Proof** Consider a set of points $x'_1, \ldots, x'_{m'}$, $m' \in \mathbb{N}, x'_i \in \mathcal{X}$, and let $K$ be the $(m + m') \times (m + m')$ Gram matrix based on $x_1, \ldots, x_m, x'_1, \ldots, x'_{m'}$. Apply Proposition 7 using $c_{m+1} = \ldots = c_{m+m'} = 0$. ■

**Example 9 (SVMs and kernel PCA)** *(i) The above results show that conditionally positive definite kernels are a natural choice whenever we are dealing with a translation invariant problem, such as the SVM: maximization of the margin of separation between two classes of data is independent of the origin's position. Seen in this light, it is not surprising that the structure of the dual optimization problem (cf. [13]) allows cpd kernels: as noticed in [11, 10], the constraint $\sum_{i=1}^m \alpha_i y_i = 0$ projects out the same subspace as (6) in the definition of conditionally positive definite kernels.*

*(ii) Another example of a kernel algorithm that works with conditionally positive definite kernels is kernel PCA [9], where the data is centered, thus removing the dependence on the origin in feature space. Formally, this follows from Proposition 7 for $c_i = 1/m$.*

**Example 10 (Parzen windows)** *One of the simplest distance-based classification algorithms conceivable proceeds as follows. Given $m_+$ points labelled with $+1$, $m_-$ points labelled with $-1$, and a test point $\phi(x)$, we compute the mean squared distances between the latter and the two classes, and assign it to the one where this mean is smaller,*

$$y = sgn\left(\frac{1}{m_-}\sum_{y_i=-1}\|\phi(x)-\phi(x_i)\|^2 - \frac{1}{m_+}\sum_{y_i=1}\|\phi(x)-\phi(x_i)\|^2\right). \quad (18)$$

*We use the distance kernel trick (Proposition 5) to express the decision function as a kernel expansion in input space: a short calculation shows that*

$$y = sgn\left(\frac{1}{m_+}\sum_{y_i=1}k(x,x_i) - \frac{1}{m_-}\sum_{y_i=-1}k(x,x_i) + c\right), \quad (19)$$

*with the constant offset $c = (1/2m_-)\sum_{y_i=-1}k(x_i,x_i)-(1/2m_+)\sum_{y_i=1}k(x_i,x_i)$. Note that for some cpd kernels, such as (8), $k(x_i,x_i)$ is always 0, thus $c = 0$. For others, such as the commonly used Gaussian kernel, $k(x_i,x_i)$ is a nonzero constant, in which case $c$ also vanishes.*

*For normalized Gaussians and other kernels that are valid density models, the resulting decision boundary can be interpreted as the Bayes decision based on two Parzen windows density estimates of the classes; for general cpd kernels, the analogy is a mere formal one.*

**Example 11 (Toy experiment)** *In Fig. 1, we illustrate the finding that kernel PCA can be carried out using cpd kernels. We use the kernel (8). Due to the centering that is built into kernel PCA (cf. Example 9, (ii), and (5)), the case $\beta = 2$ actually is equivalent to linear PCA. As we decrease $\beta$, we obtain increasingly nonlinear feature extractors.*

*Note, moreover, that as the kernel parameter $\beta$ gets smaller, less weight is put on large distances, and we get more localized feature extractors (in the sense that the regions where they have large gradients, i.e. dense sets of contour lines in the plot, get more localized).*

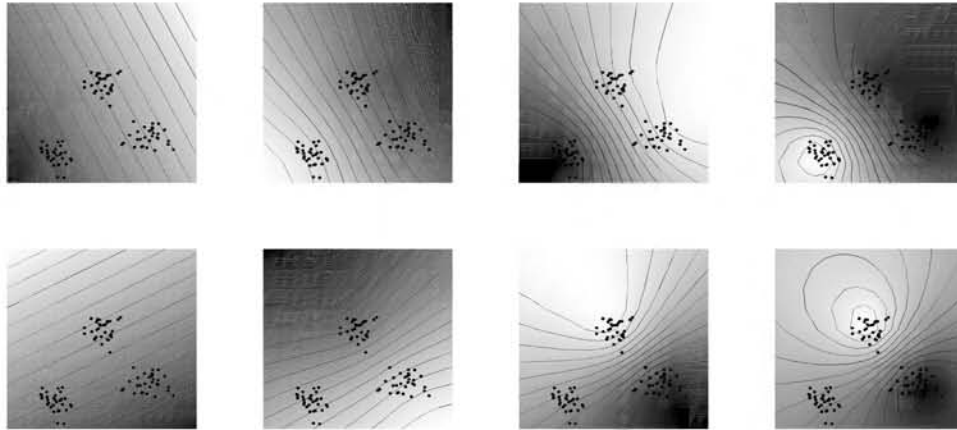

Figure 1: Kernel PCA on a toy dataset using the cpd kernel (8); contour plots of the feature extractors corresponding to projections onto the first two principal axes in feature space. From left to right: $\beta = 2, 1.5, 1, 0.5$. Notice how smaller values of $\beta$ make the feature extractors increasingly nonlinear, which allows the identification of the cluster structure.

# 3 Conclusion

We have described a kernel trick for distances in feature spaces. It can be used to generalize all distance based algorithms to a feature space setting by substituting a suitable kernel function for the squared distance. The class of kernels that can be used is larger than those commonly used in kernel methods (known as positive definite kernels). We have argued that this reflects the translation invariance of distance based algorithms, as opposed to genuinely dot product based algorithms. SVMs and kernel PCA are translation invariant in feature space, hence they are really both distance rather than dot product based. We thus argued that they can both use conditionally positive definite kernels. In the case of the SVM, this drops out of the optimization problem automatically [11], in the case of kernel PCA, it corresponds to the introduction of a reference point in feature space. The contribution of the present work is that it identifies translation invariance as the underlying reason, thus enabling us to use cpd kernels in a much larger class of kernel algorithms, and that it draws the learning community's attention to the kernel trick for distances.

**Acknowledgments.** Part of the work was done while the author was visiting the Australian National University. Thanks to Nello Cristianini, Ralf Herbrich, Sebastian Mika, Klaus Müller, John Shawe-Taylor, Alex Smola, Mike Tipping, Chris Watkins, Bob Williamson, Chris Williams and a conscientious anonymous reviewer for valuable input.

# References

[1] M. A. Aizerman, E. M. Braverman, and L. I. Rozonoér. Theoretical foundations of the potential function method in pattern recognition learning. *Autom. and Remote Contr.*, 25:821–837, 1964.

[2] C. Berg, J.P.R. Christensen, and P. Ressel. *Harmonic Analysis on Semigroups*. Springer-Verlag, New York, 1984.

[3] B. E. Boser, I. M. Guyon, and V. N. Vapnik. A training algorithm for optimal margin classifiers. In D. Haussler, editor, *Proceedings of the 5th Annual ACM Workshop on Computational Learning Theory*, pages 144–152, Pittsburgh, PA, July 1992. ACM Press.

[4] F. Girosi, M. Jones, and T. Poggio. Regularization theory and neural networks architectures. *Neural Computation*, 7(2):219–269, 1995.

[5] D. Haussler. Convolutional kernels on discrete structures. Technical Report UCSC-CRL-99-10, Computer Science Department, University of California at Santa Cruz, 1999.

[6] J. Mercer. Functions of positive and negative type and their connection with the theory of integral equations. *Philos. Trans. Roy. Soc. London*, A 209:415–446, 1909.

[7] I. J. Schoenberg. Metric spaces and positive definite functions. *Trans. Amer. Math. Soc.*, 44:522–536, 1938.

[8] B. Schölkopf, C. J. C. Burges, and A. J. Smola. *Advances in Kernel Methods — Support Vector Learning*. MIT Press, Cambridge, MA, 1999.

[9] B. Schölkopf, A. Smola, and K.-R. Müller. Nonlinear component analysis as a kernel eigenvalue problem. *Neural Computation*, 10:1299–1319, 1998.

[10] A. Smola, T. Frieß, and B. Schölkopf. Semiparametric support vector and linear programming machines. In M.S. Kearns, S.A. Solla, and D.A. Cohn, editors, *Advances in Neural Information Processing Systems 11*, pages 585 – 591, Cambridge, MA, 1999. MIT Press.

[11] A. Smola, B. Schölkopf, and K.-R. Müller. The connection between regularization operators and support vector kernels. *Neural Networks*, 11:637–649, 1998.

[12] W.S. Torgerson. *Theory and Methods of Scaling*. Wiley, New York, 1958.

[13] V. Vapnik. *The Nature of Statistical Learning Theory*. Springer, N.Y., 1995.

[14] G. Wahba. *Spline Models for Observational Data*, volume 59 of *CBMS-NSF Regional Conference Series in Applied Mathematics*. SIAM, Philadelphia, 1990.

[15] C. Watkins, 2000. personal communication.
